# A Massively-Parallel SIMD Processor for Neural Network and Machine Vision Applications

**Michael A. Glover**
Current Technology, Inc.
99 Madbury Road
Durham, NH 03824

**W. Thomas Miller, III**
Department of Electrical and Computer Engineering
The University of New Hampshire
Durham, NH 03824

## Abstract

This paper describes the MM32k, a massively-parallel SIMD computer which is easy to program, high in performance, low in cost and effective for implementing highly parallel neural network architectures. The MM32k has 32768 bit serial processing elements, each of which has 512 bits of memory, and all of which are interconnected by a switching network. The entire system resides on a single PC-AT compatible card. It is programmed from the host computer using a C++ language class library which abstracts the parallel processor in terms of fast arithmetic operators for vectors of variable precision integers.

## 1  INTRODUCTION

Many well known neural network techniques for adaptive pattern classification and function approximation are inherently highly parallel, and thus have proven difficult to implement for real-time applications at a reasonable cost. This includes

a variety of learning systems such as radial basis function networks [Moody 1989], Kohonen self-organizing networks [Kohonen 1982], ART family networks [Carpenter 1988], and nearest-neighbor interpolators [Duda 1973], among others. This paper describes the MM32k, a massively-parallel SIMD computer which is easy to program, high in performance, low in cost and effective for implementing highly parallel neural network architectures. The MM32k acts as a coprocessor to accelerate vector arithmetic operations on PC-AT class computers, and can achieve giga-operation per second performance on suitable problems. It is programmed from the host computer using a C++ language class library, which overloads typical arithmetic operators, and supports variable precision arithmetic. The MM32k has 32768 bit serial PEs, or processing elements, each of which has 512 bits of memory, and all of which are interconnected by a switching network. The PEs are combined with their memory on an single DRAM memory chip giving 2048 processors per chip. The entire 32768 processor system resides on a single ISA bus compatible card. It is much more cost effective than other SIMD processors [Hammerstrom 1990; Hillis 1985; Nickolls 1990; Potter 1985] and more flexible than fixed purpose chips [Holler 1991].

## 2   SIMD ARCHITECTURE

The SIMD PE array contains 32768 one bit processors, each with 512 bits of memory and a connection to the interconnection network. The PE array design is unique in that 2048 PEs, including their PE memory, are realized on a single chip. The total PE array memory is 2 megabytes and has a peak memory bandwidth is 25 gigabytes per second. The PE array can add 8 bit integers at 2.5 gigaoperations per second. It also dissipates less than 10 watts of power and is shown in Figure 1.

Each PE has three one bit registers, a 512 bit memory, and a one bit ALU. It performs bit serial arithmetic and can therefore vary the number of bits of precision to fit the problem at hand, saving SIMD instruction cycles and SIMD memory. There are 17 instructions in the PE instruction set, all of which execute at a 6.25 MIPS rate. The PE instruction set is functionally complete in that it can perform boolean NOT and OR functions and can therefore perform any operation, including arithmetic and conditional operations. A single PE is shown in Figure 2.

The interconnection network allows data to be sent from one PE to another. It is implemented by a 64*64 full crossbar switch with 512 PEs connected to each port of the switch. It allows data to be sent from one PE to another PE, an arbitrary distance away, in constant time. The peak switch bandwidth is 280 megabytes per second. The switch also allows the PE array to perform data reduction operations, such as taking the sum or maximum over data elements distributed across all PEs.

## 3   C++ PROGRAMMING ENVIRONMENT

The purpose of the C++ programming environment is to allow a programmer to declare and manipulate vectors on the MM32k as if they were variables in a program running on the host computer. Programming is performed entirely on the host, using standard MS-DOS or Windows compatible C++ compilers. The C++ programming environment for the MM32k is built around a C++ class, named

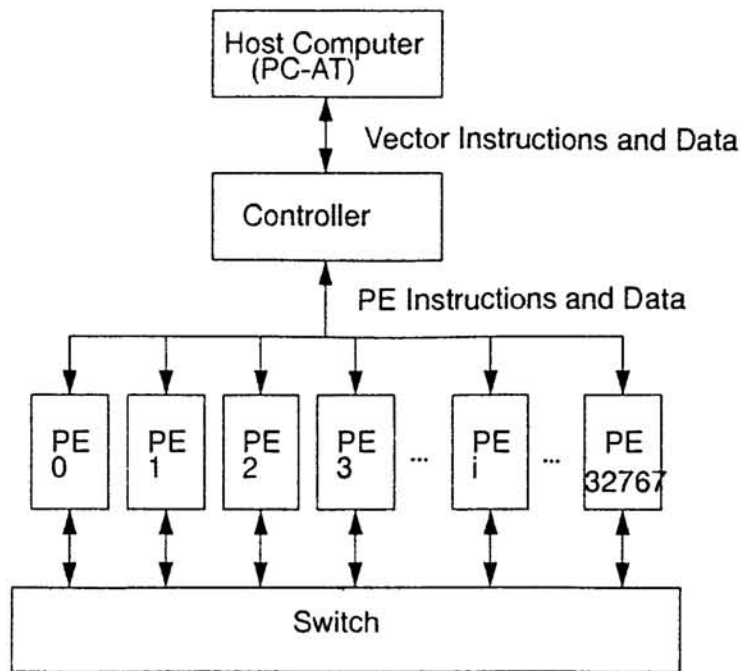

Figure 1: A block diagram of the MM32k.

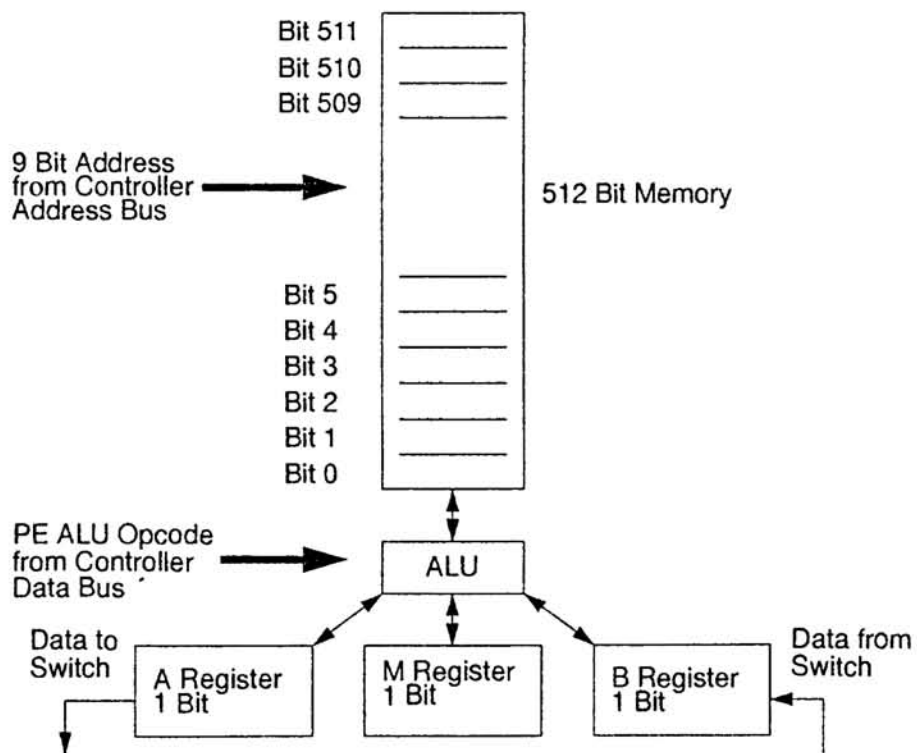

Figure 2: A block diagram of a single processing element (PE).

Table 1: 8 Bit Operations With 32768 and 262144 Elements

| 8 bit operation | Actual MOPS with length of 32768 | Actual MOPS with length of 262144 |
| --- | --- | --- |
| copy | 1796 | 9429 |
| vector+vector | 1455 | 2074 |
| vector+scalar | 1864 | 3457 |
| vector*vector | 206 | 215 |
| vector*scalar | 426 | 450 |
| vector>scalar | 1903 | 6223 |
| align(vector,scalar) | 186 | 213 |
| sum(vector) | 52 | 306 |
| maximum(vector) | 114 | 754 |

MM_VECTOR, which represents a vector of integers. Most of the standard C arithmetic operators, such as +, -, *, /, =, and > have been overloaded to work with this class. Some basic functions, such as absolute value, square root, minimum, maximum, align, and sum, have also been overloaded or defined to work with the class.

The significance of the class MM_VECTOR is that instances of it look and act like ordinary variables in a C++ program. So a programmer may add, subtract, assign, and manipulate these vector variables from a program running on the host computer, but the storage associated with them is in the SIMD memory and the vector operations are performed in parallel by the SIMD PEs. MM_VECTORs can be longer than 32768. This is managed (transparent to the host program) by placing two or more vector elements in the SIMD memory of each PE. The class library keeps track of the number of words per PE. MM_VECTORs can be represented by different numbers of bits. The class library automatically keeps track of the number of bits needed to represent each MM_VECTOR without overflow. For example, if two 12 bit integers were added together, then 13 bits would be needed to represent the sum without overflow. The resulting MM_VECTOR would have 13 bits. This saves SIMD memory space and SIMD PE instruction cycles. The performance of the MM32k on simple operators running under the class library is listed in Table 1.

## 4  NEURAL NETWORK EXAMPLES

A common operation found in neural network classifiers (Kohonen, ART, etc.) is the multi-dimensional nearest-neighbor match. If the network has a large number of nodes, this operation is particularly inefficient on single processor systems, which must compute the distance metric for each node sequentially. Using the MM32k, the distance metrics for all nodes (up to 32768 nodes) can be computed simultaneously, and the identification of the minimum distance can be made using an efficient tree compare included in the system microcode.

Table 2: Speedup on Nearest Neighbor Search

| Processor | Time for 32768 nodes | Time for 65536 nodes | MM32k speedup for 32768 nodes | MM32k speedup for 65536 nodes |
|---|---|---|---|---|
| MM32k | 2.2 msec | 3.1 msec | 1:1 | 1:1 |
| i486 | 350 msec | 700 msec | 159:1 | 226:1 |
| MIPS | 970 msec | 1860 msec | 441:1 | 600:1 |
| Alpha | 81 msec | 177 msec | 37:1 | 57:1 |
| SPARC | 410 msec | 820 msec | 186:1 | 265:1 |

Figure 3 shows a C++ code example for performing a 16-dimensional nearest neighbor search over 32768 nodes. The global MM_VECTOR variable state[16] defines the 16-dimensional location of each node. Each logical element of state[ ] (state[0], state[1], etc.) is actually a vector with 32768 elements distributed across all processors. The routine find_best_match() computes the euclidean distance between each node's state and the current test vector test_input[ ], which resides on the host processor. Note that the equations appear to be scalar in nature, but in fact direct vector operations to be performed by all processors simultaneously.

The performance of the nearest neighbor search shown in Figure 3 is listed in Table 2. Performance on the same task is also listed for four comparison processors: a Gateway2000 model 4DX2-66V with 66 MHz 80486 processor (i486), a DECstation 5000 Model 200 with 25 MHz MIPS R3000A processor (MIPS), a DECstation 3000 Model 500AXP with 150 MHz Alpha AXP processor (Alpha), and a Sun SPARC-station 10 Model 30 with 33 MHz SuperSPARC processor (SPARC). There are 16 subtractions, 16 additions, 16 absolute values, one global minimum, and one global first operation performed. The MM32k is tested on problems with 32768 and 65536 exemplars and compared against four popular serial machines performing equivalent searches. The MM32k requires 3.1 milliseconds to search 65536 exemplars which is 265 times faster than a SPARC 10.

The flexibility of the MM32k for neural network applications was demonstrated by implementing complete fixed-point neural network paradigms on the MM32k and on the four comparison processors (Table 3). Three different neural network examples were evaluated. The first was a radial basis function network with 32,768 basis functions (rational function approximations to gaussian functions). Each basis function had 9 8-bit inputs, 3 16-bit outputs (a vector basis function magnitude), and independent width parameters for each of the nine inputs. The performances listed in the table (RBF) are for feedforward response only. The second example was a Kohonen self-organizing network with a two-dimensional sheet of Kohonen nodes of dimension 200x150 (30,000 nodes). The problem was to map a nonlinear robotics forward kinematics transformation with eight degrees of freedom (8-bit parameters) onto the two-dimensional Kohonen layer. The performances listed in the table (Kohonen) are for self-organizing training. The third example problem was a neocognitron for target localization in a 256x256 8-bit input image. The first hidden layer of the neocognitron had 8 256x256 sheets of linear convolution units

```
/* declare 16-D MM32k exemplars */
MM_VECTOR state[16] = {
    MM_VECTOR(32768), MM_VECTOR(32768),
    MM_VECTOR(32768), MM_VECTOR(32768),
    MM_VECTOR(32768), MM_VECTOR(32768),
    MM_VECTOR(32768), MM_VECTOR(32768),
    MM_VECTOR(32768), MM_VECTOR(32768),
    MM_VECTOR(32768), MM_VECTOR(32768),
    MM_VECTOR(32768), MM_VECTOR(32768),
    MM_VECTOR(32768), MM_VECTOR(32768)
};

/* return PE number of processor with closest match */
long find_best_match(long test_input[16])
{
    int i;
    MM_VECTOR difference(32768);   /* differences */
    MM_VECTOR distance(32768);     /* distances   */

    /* compute the 16-D distance scores */
    distance = 0;
    for (i=0; i<16; ++i)  {
        difference = state[i] - test_input[i];
        distance = distance + (difference * difference);
    }

    /* return the PE number for minimum distance */
    return first(distance == minimum(distance));
}
```

Figure 3: A C++ code example implementing a nearest neighbor search.

Table 3: MM32k Speedup for Select Neural Network Paradigms

| Processor | RBF | Kohonen | NCGTRN |
|---|---|---|---|
| MM32k | 1:1 | 1:1 | 1:1 |
| i486 | 161:1 | 76:1 | 336:1 |
| MIPS | 180:1 | 69:1 | 207:1 |
| Alpha | 31:1 | 11:1 | 35:1 |
| SPARC | 94:1 | 49:1 | 378:1 |

with 16x16 receptive fields in the input image. The second hidden layer of the neocognitron had 8 256x256 sheets of sigmoidal units (fixed-point rational function approximations to sigmoid functions) with 3x3x8 receptive fields in the first hidden layer. The output layer of the neocognitron had 256x256 sigmoidal units with 3x3x8 receptive fields in the second hidden layer. The performances listed in the table (NCGTRN) correspond to feedforward response followed by backpropagation training. The absolute computation times for the MM32k were 5.1 msec, 10 msec, and 1.3 sec, for the RBF, Kohonen, and NCGTRN neural networks, respectively.

## Acknowledgements

This work was supported in part by a grant from the Advanced Research Projects Agency (ARPA/ONR Grant #N00014-92-J-1858).

## References

J. L. Potter. (1985) *The Massively Parallel Processor*, Cambridge, MA: MIT Press.

G. A. Carpenter and S. Grossberg. (1988) The ART of adaptive pattern recognition by a self-organizing neural network. *Computer* vol. 21, pp. 77-88.

R. O. Duda and P. E. Hart. (1973) *Pattern Classification and Scene Analysis*. New York: Wiley.

D. Hammerstrom. (1990) A VLSI architecture for high-performance, low cost, on-chip learning, in Proc. IJCNN, San Diego, CA, June 17-21, vol. II, pp. 537-544.

W. D. Hillis. (1985) *The Connection Machine*. Cambridge, MA: MIT Press.

M. Holler. (1991) VLSI implementations of learning and memory systems: A review. In *Advances in Neural Information Processing Systems 3*, ed. by R. P. Lippman, J. E. Moody, and D. S. Touretzky, San Francisco, CA: Morgan Kaufmann.

T. Kohonen. (1982) Self-organized formation of topologically correct feature maps. *Biological Cybernetics*, vol. 43, pp. 56-69.

J. Moody and C. Darken. (1989) Fast learning in networks of locally- tuned processing units. *Neural Computation*, vol. 1, pp. 281-294.

J. R. Nickolls. (1990) The design of the MasPar MP-1: A cost-effective massively parallel computer. In Proc. COMPCON Spring '90, San Francisco, CA, pp. 25-28..